# Simulation of a Thalamocortical Circuit for Computing Directional Heading in the Rat

**Hugh T. Blair***
Department of Psychology
Yale University
New Haven, CT 06520-8205
tadb@minerva.cis.yale.edu

## Abstract

Several regions of the rat brain contain neurons known as *head-direction cells*, which encode the animal's directional heading during spatial navigation. This paper presents a biophysical model of head-direction cell activity, which suggests that a thalamocortical circuit might compute the rat's head direction by integrating the angular velocity of the head over time. The model was implemented using the neural simulator NEURON, and makes testable predictions about the structure and function of the rat head-direction circuit.

## 1 HEAD-DIRECTION CELLS

As a rat navigates through space, neurons called *head-direction cells* encode the animal's directional heading in the horizontal plane (Ranck, 1984; Taube, Muller, & Ranck, 1990). Head-direction cells have been recorded in several brain areas, including the postsubiculum (Ranck, 1984) and anterior thalamus (Taube, 1995). A variety of theories have proposed that head-direction cells might play an important role in spatial learning and navigation (Brown & Sharp, 1995; Burgess, Recce, & O'Keefe, 1994; McNaughton, Knierim, & Wilson, 1995; Wan, Touretzky, & Redish, 1994; Zhang, 1995).

### 1.1 BASIC FIRING PROPERTIES

A head-direction cell fires action potentials only when the rat's head is facing in a particular direction with respect to the static surrounding environment, regardless of the animal's location within that environment. Head-direction cells are *not* influenced by the position of the rat's head with respect to its body, they are only influenced by the direction of the

*Also at the Yale Neuroengineering and Neuroscience Center (NNC), 5 Science Park North, New Haven, CT 06511

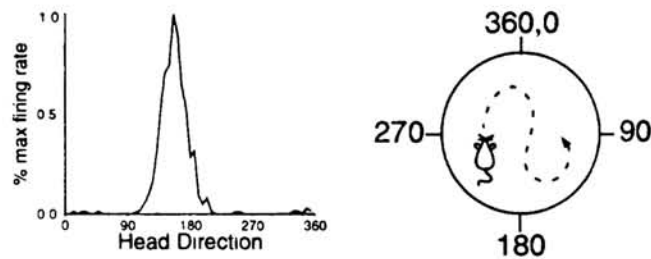

Figure 1: Directional Tuning Curve of a Head-Direction Cell

head with respect to the stationary reference frame of the spatial environment. Each head-direction cell has its own directional preference, so that together, the entire population of cells can encode any direction that the animal is facing.

Figure 1 shows an example of a head-direction cell's *directional tuning curve*, which plots the firing rate of the cell as a function of the rat's momentary head direction. The tuning curve shows that this cell fires maximally when the rat's head is facing in a preferred direction of about 160 degrees. The cell fires less rapidly for directions close to 160 degrees, and stops firing altogether for directions that are far from 160 degrees.

## 1.2 THE VELOCITY INTEGRATION HYPOTHESIS

McNaughton, Chen, & Markus (1991) have proposed that head-direction cells might rely on a process of *dead-reckoning* to calculate the rat's current head direction, based on the previous head direction and the angular velocity at which the head is turning. That is, head-direction cells might compute the directional position of the head by integrating the angular velocity of the head over time. This velocity integration hypothesis is supported by three experimental findings. First, several brain regions that are associated with head-direction cells contain *angular velocity cells*, neurons that fire in proportion to the angular head velocity (McNaughton et al., 1994; Sharp, in press). Second, some head-direction cells in postsubiculum are modulated by angular head velocity, such that their peak firing rate is higher if the head is turning in one direction than in the other (Taube et al., 1990). Third, it has recently been found that head-direction cells in the anterior thalamus, but not the postsubiculum, anticipate the *future* direction of the rat's head (Blair & Sharp, 1995).

## 1.3 ANTICIPATORY HEAD-DIRECTION CELLS

Blair and Sharp (1995) discovered that head-direction cells in the anterior thalamus shift their directional preference to the left during clockwise turns, and to the right during counterclockwise turns. They showed that this shift occurs systematically as a function of head velocity, in a way that allows these cells anticipate the future direction of the rat's head. To illustrate this, consider a cell that fires whenever the head will be facing a specific direction, $\theta$, in the near future. How would such a cell behave? There are three cases to consider. First, imagine that the rat's head is turning clockwise, approaching the direction $\theta$ from the left side. In this case, the anticipatory cell must fire when the head is facing to the left of $\theta$, because being to the left of $\theta$ and turning clockwise predicts arrival at $\theta$ in the near future. Second, when the head is turning counterclockwise and approaching $\theta$ from the right side, the anticipatory cell must fire when the head is to the right of $\theta$. Third, if the head is still, then the cell should only fire if the head is presently facing $\theta$.

In summary, an anticipatory head direction cell should shift its directional preference to the left during clockwise turns, to the right during counterclockwise turns, and not at all when the head is still. This behavior can be formalized by the equation

$$\mu(v) = \theta - v\tau,$$  [1]

where μ denotes the cell's preferred present head direction, ν denotes the angular velocity of the head, θ denotes the future head direction that the cell anticipates, and τ is a constant time delay by which the cell's activity anticipates arrival at θ. Equation 1 assumes that μ is measured in degrees, which increase in the clockwise direction, and that ν is positive for clockwise head turns, and negative for counterclockwise head turns. Blair & Sharp (1995) have demonstrated that Equation 1 provides a good approximation of head-direction cell behavior in the anterior thalamus.

## 1.3  ANTICIPATORY TIME DELAY (τ)

Initial reports suggested that head-direction cells in the anterior thalamus anticipate the future head direction by an average time delay of τ = 40 msec, whereas postsubicular cells encode the present head direction, and therefore "anticipate" by τ = 0 msec (Blair & Sharp, 1995; Taube & Muller, 1995). However, recent evidence suggests that individual neurons in the anterior thalamus may be temporally tuned to anticipate the rat's future head-direction by different time delays between 0-100 msec, and that postsubicular cells may "lag behind" the present head-direction by about 10 msec (Blair & Sharp, 1996).

## 2  A BIOPHYSICAL MODEL

This section describes a biophysical model that accounts for the properties of head-direction cells in postsubiculum and anterior thalamus, by proposing that they might be connected to form a thalamocortical circuit. The next section presents simulation results from an implementation of the model, using the neural simulator NEURON (Hines, 1993).

## 2.1  NEURAL ELEMENTS

Figure 2 illustrates a basic circuit for computing the rat's head-direction. The circuit consists of five types of cells: 1) *Present Head-Direction* (PHD) Cells encode the present direction of the rat's head, 2) *Anticipatory Head-Direction* (AHD) Cells encode the future direction of the rat's head, 3) *Angular-Velocity* (AV) Cells encode the angular velocity of the rat's head (the CLK AV Cell is active during clockwise turns, and the CNT AV Cell is active during counterclockwise turns), 4) the *Angular Speed* (AS) Cell fires in inverse proportion to the angular speed of the head, regardless of the turning direction (that is, the AS Cell fires at a lower rate during fast turns, and at a higher rate during slow turns), 5) *Angular-Velocity Modulated Head-Direction* (AVHD) Cells are head-direction cells that fire

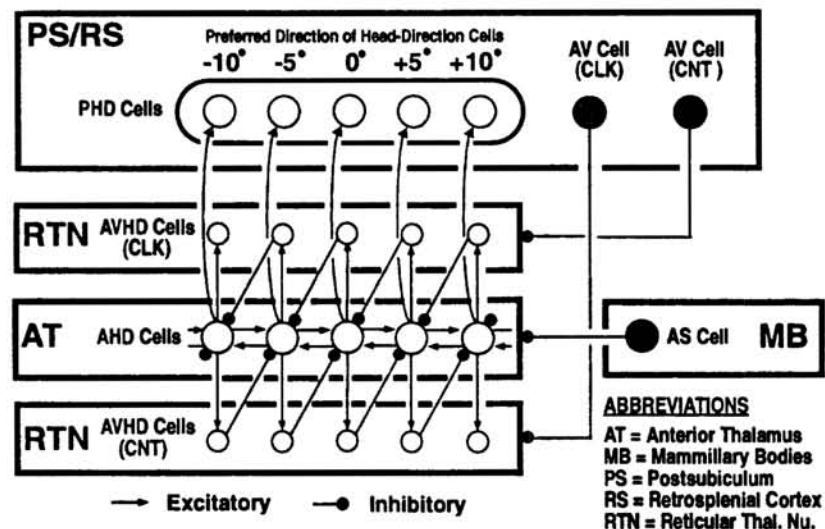

Figure 2: A Model of the Rat Head-Direction System

only when the head is turning in one direction and not the other (the CLK AVHD Cell fires in its preferred direction only when the head is turning clockwise, and the CNT AVHD Cell fires in its preferred direction only when the head turns counterclockwise).

## 2.2 FUNCTIONAL CHARACTERISTICS

In the model, AHD Cells directly excite their neighbors on either side, but indirectly inhibit these same neighbors via the AVHD Cells, which act as inhibitory interneurons. AHD Cells also send excitatory feedback connections to themselves (omitted from Figure 2 for clarity), so that once they become active, they remain active until they are turned off by inhibitory input (the rate of firing can also be modulated by inhibitory input). When the rat is not turning its head, the cell representing the current head direction fires constantly, both exciting and inhibiting its neighbors. In the steady-state condition (i.e., when the rat is not turning its head), lateral inhibition exceeds lateral excitation, and therefore activity does not spread in either direction through the layer of AHD Cells. However, when the rat begins turning its head, some of the AVHD Cells are turned off, allowing activity to spread in one direction. For example, during a clockwise head turn, the CLK AV Cell becomes active, and inhibits the layer of CNT AVHD Cells. As a result, AHD Cells stop inhibiting their right neighbors, so activity spreads to the right through the layer of AHD Cells. Because AHD Cells continue to inhibit their neighbors to the left, activity is shut down in the leftward direction, in the wake of the activity spreading to the right.

The speed of propagation through the AHD layer is governed by the AS Cell. During slow head turns, the AS Cell fires at a high rate, strongly inhibiting the AHD Cells, and thereby slowing the speed of propagation. During fast head turns, the AS Cell fires at a low rate, weakly inhibiting the AHD Cells, allowing activity to propagate more quickly. Because of inhibition from AS cells, AHD cells fire faster when the head is turning than when it is still (see Figure 4), in agreement with experimental data (Blair & Sharp, 1995).

AHD Cells send a topographic projection to PHD Cells, such that each PHD Cell receives excitatory input from an AHD Cell that anticipates when the head will soon be facing in the PHD Cell's preferred direction. AHD Cell activity anticipates PHD Cell activity because there is a transmission delay between the AHD and PHD Cells (assumed to be 5 msec in the simulations presented below). Also, the weights of the connections from AHD Cells to PHD Cells are small, so each AHD Cell must fire several action potentials before its targeted PHD Cell can begin to fire. The time delay between AHD and PHD Cells accounts for anticipatory firing, and corresponds to the $\tau$ parameter in Equation 1.

## 2.3 ANATOMICAL CHARACTERISTICS

Each component of the model is assumed to reside in a specific brain region. AHD and PHD Cells are assumed to reside in anterior thalamus (AT) and postsubiculum (PS), respectively. AS Cells have been observed in PS (Sharp, in press) and retrosplenial cortex (RS) (McNaughton, Green, & Mizumori, 1986), but the model predicts that they may also be found in the mammillary bodies (MB), since MB receives input from PS and RS (Shibata, 1989), and MB projects to ATN. AVHD Cells have been observed in PS (Taube et al., 1990), but the model predicts that they may also be found in the reticular thalamic nucleus (RTN), because RTN receives input from PS/RS (Lozsadi, 1994), and RTN inhibits AT. It should be noted that lateral excitation between ATN cells has not been shown, so this feature of the model may be incorrect. Table 1 summarizes anatomical evidence.

## 3  SIMULATION RESULTS

The model illustrated in Figure 2 has been implemented using the neural simulator NEURON (Hines, 1993). Each neural element was represented as a single spherical compart-

Table 1: Anatomical Features of the Model

| FEATURE OF MODEL | REFERENCE |
|---|---|
| PHD Cells in PS/RS | Chen et al., 1990; Ranck, 1984 |
| AHD Cells in AT | Blair & Sharp, 1995 |
| AV Cells in PS/RS | McNaughton et al., 1994; Sharp, in press |
| AT projects to PS | van Groen & Wyss, 1990 |
| AT projects to RTN | Shibata, 1992 |
| PS/RS projects to RTN | Lozsadi, 1994 |
| AVHD Cells in RTN | PREDICTION OF MODEL |
| AS Cells in MB | PREDICTION OF MODEL |

ment, 30 μm in diameter, with RC time constants ranging between 15 and 30 msec. Synaptic connections were simulated using triggered alpha-function conductances. The results presented here demonstrate the behavior of the model, and compare the properties of the model with experimental data.

To begin each simulation, a small current was injected in to one of the AHD Cells, causing it to initiate sustained firing. This cell represented the simulated rat's initial head direction. Head-turning behavior was simulated by injecting current into the AV and AS Cells, with an amplitude that yielded firing proportional to the desired angular head velocity.

## 3.1 ACTIVITY OF HEAD-DIRECTION CELLS

Figure 3 presents a simple simulation, which illustrates the behavior of head-direction cells in the model. The simulated rat begins by facing in the direction of 0 degrees. Over the course of 250 msec, the rat quickly turns its head 60 degrees to the right, and then returns to the initial starting position of 0 degrees. The average velocity of the head in this simulation was 480 degrees/sec, which is similar to the speed at which an actual rat performs a fast head turn (Blair & Sharp, 1995). Over the course of the simulation, neural activation propagates from the 0-degree cell to the 60-degree cell, and then back to the 0-degree cell.

## 3.2 COMPARISON WITH EXPERIMENTAL DATA

To examine how well the model reproduces firing properties of PS and AT cells, another simple simulation was performed. The firing rate the model's PHD and AHD Cells was examined while the simulated rat performed several 360-degree revolutions in both the clockwise and counterclockwise directions. Results are summarized in Figure 4, which

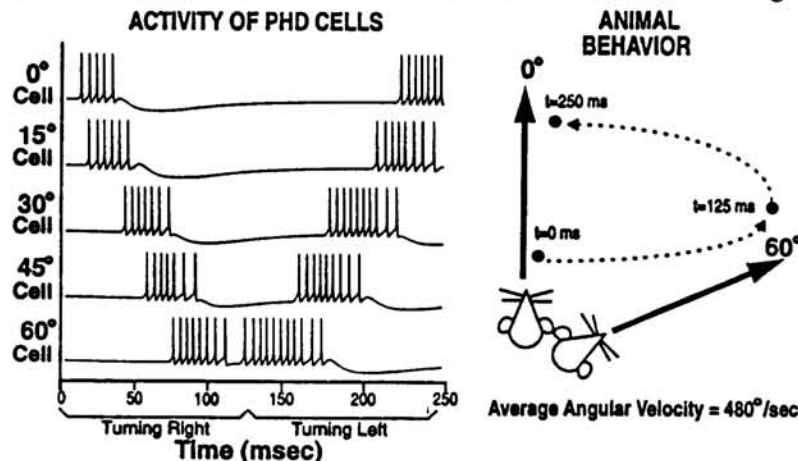

Figure 3: Simulation Example

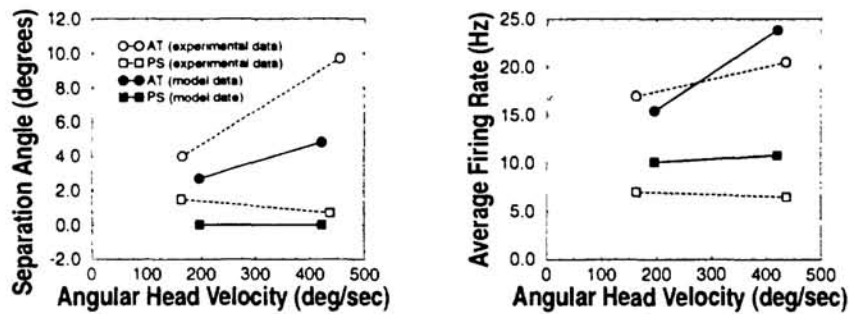

Figure 4: Compared Properties of Real and Simulated Head-Direction Cells

compares simulation data with experimental data. The experimental data in Figure 4 shows averaged results for 21 cells recorded in AT, and 19 cells recorded in PS.

Because AT cells anticipate the future head direction, they exhibit an angular separation between their clockwise and counterclockwise directional preference, whereas as no such separation occurs for PS cells (see section 2.4). For AT cells, the magnitude of the angular separation is proportional to angular head velocity, with greater separation occurring for fast turns, and less separation for slow turns (see Eq. 1). The left panel of Figure 4 shows that the model's PHD and AHD Cells exhibit a similar pattern of angular separation.

Blair & Sharp (1995) reported that the firing rates of AT and PS cells differ in two ways: 1) AT cells fire at a higher rate than PS cells, and 2) AT cells have a higher rate during fast turns than during slow turns, whereas PS cells fire at the same rate, regardless of turning speed. In Figure 4 (right panel), it can be seen that the model reproduces these findings.

# 4 DISCUSSION AND CONCLUSIONS

In this paper, I have presented a neural model of the rat head-direction system. The model includes neural elements whose firing properties are similar to those of actual neurons in the rat brain. The model suggests that a thalamocortical circuit might compute the directional position of the rat's head, by integrating angular head velocity over time.

## 4.1 COMPARISON WITH OTHER MODELS

McNaughton et al. (1991) proposed that neurons encoding head-direction and angular velocity might be connected to form a linear associative mapping network. Skaggs et al. (1995) have refined this idea into a theoretical circuit, which incorporates head-direction and angular velocity cells. However, the Skaggs et al. (1995) circuit does not incorporate anticipatory head-direction cells, like those found in AT. A model that does incorporate anticipatory cells has been developed by Elga, Redish, & Touretzky (unpublished manuscript). Zhang (1995) has recently presented a theoretical analysis of the head-direction circuit, which suggests that anticipatory head-direction cells might be influenced by both the angular velocity and angular acceleration of the head, whereas non-anticipatory cells may be influenced by the angular velocity only, and not the angular acceleration.

## 4.2 LIMITATIONS OF THE MODEL

In its current form, the model suffers some significant limitations. For example, the directional tuning curves of the model's head-direction cells are much narrower than those of actual head-direction cells. Also, in its present form, the model can accurately track the rat's head-direction over a rather limited range of angular head velocities. These limitations are presently being addressed in a more advanced version of the model.

**Acknowledgments**

This work was supported by NRSA fellowship number 1 F31 MH11102-01A1 from NIMH, a Yale Fellowship, and the Yale Neuroengineering and Neuroscience Center (NNC). I thank Michael Hines, Patricia Sharp, and Steve Fisher for their assistance.

**References**

Blair, H.T., & Sharp, P.E. (1995). Anticipatory head-direction cells in anterior thalamus: Evidence for a thalamocortical circuit that integrates angular head velocity to compute head direction. *Journal of Neuroscience, 15,* 6260-6270.

Blair, H.T., & Sharp (1996). Temporal Tuning of Anticipatory Head-Direction Cells in the Anterior Thalamus of the Rat. *Submitted.*

Brown, M. & Sharp, P.E. (1995). Simulation of spatial learning in the morris water maze by a neural network model of the hippocampal formation and nucleus accumbens. *Hippocampus, 5,* 171-188.

Burgess, N., Recce, M., & O'Keefe, J. (1994). A model of hippocampal function. *Neural Networks, 7,* 1065-1081.

Elga, A.N., Redish, A.D., & Touretzky, D.S. (1995). A model of the rodent head-direction system. *Unpublished Manuscript.*

Hines, M. (1993). NEURON: A program for simulation of nerve equations. In F. Eckman (Ed.), *Neural Systems: Analysis and Modeling,* Norwell, MA : Kluwer Academic Publishers, pp. 127-136.

Lozsadi, D.A. (1994). Organization of cortical afferents to the rostral, limbic sector of the rat thalamic reticular nucleus. *The Journal of Comparative Neurology, 341,* 520-533.

McNaughton, B.L., Chen, L.L., & Markus, E.J. (1991). Dead reckoning, landmark learning, and the sense of direction: a neurophysiological and computational hypothesis. *Journal of Cognitive Neuroscience, 3,* 190-202.

McNaughton, B.L., Green, E.J., & Mizumori, S.J.Y. (1986). Representation of body motion trajectory by rat sensory motor cortex neurons. *Society for Neuroscience Abstracts, 12,* 260.

McNaughton, B.L., Knierim, J.J., & Wilson, M.A. (1995). Vector encoding and the vestibular foundations of spatial cognition: neurophysiological and computational mechanisms. In M. Gazzaniga (Ed.), *The Cognitive Neurosciences.* Cambridge: MIT Press.

McNaughton, B.L., Mizumori, S.Y.J., Barnes, C.A., Leonard, B.J., Marquis, M., & Green, B.J. (1994). Coritcal representation of motion during unrestrained spatial navigaton in the rat. *Cerebral Cortex, 4,* 27-39.

Ranck, J.B. (1984). Head-direction cells in the deep cell layers of dorsal presubiculum in freely moving rats. *Society for Neuroscience Abstracts, 12,* 1524.

Shibata, H. (1989). Descending projections to the mammillary nuclei in the rat, as studied by retrograde and anterograde transport of wheat germ agglutinin-horseradish peroxidase. *The Journal of Comparative Neurology, 285,* 436-452.

Shibata, H. (1992). Topographic organization of subcortical projections to the anterior thalamic nuclei in the rat. *The Journal of Comparative Neurology, 323,* 117-127.

Sharp, P.E. (in press). Multiple spatial/behavioral corrrelates for cells in the rat postsubiculum: multiple regression analysis and comparison to other hippocampal areas. *Cerebral Cortex.*

Skaggs, W.E., Knierim, J.J., Kudrimoti, H.S., & McNaughton, B.L. (1995). A model of the neural basis of the rat's sense of direction. In G. Tesauro, D.S. Touretzky, & T.K. Leen (Eds.), *Advances in Neural Information Processing Systems 7.* MIT Press.

Taube, J.S. (1995). Head-direction cells recorded in the anterior thalamic nuclei of freely-moving rats. *Journal of Neuroscience, 15,* 70-86.

Taube, J.S., & Muller, R.U. (1995). Head-direction cell activity in the anterior thalamus, but not the postsubiculum, predicts the animal's future directional heading. *Society for Neuroscience Abstracts, 21,* 946.

Taube, J.S., Muller, R.U., & Ranck, J.B. (1990). Head-direction cells recorded from the postsubiculum in freely moving rats, I. Description and quantitative analysis. *Jounral of Neuroscience, 10,* 420-435.

van Groen, T., & Wyss, J.M. (1990). The postsubicular cortex in the rat: characterization of the fourth region of subicular cortex and its connections. *Journal of Comparative Neurology, 216,* 192-210.

Wan, H.S., Touretzky, D.S., & Redish, D.S. (1994). A rodent navigation model that combines place code, head-direction, and path integration information. *Society for Neuroscience Abstracts, 20,* 1205.

Zhang, K. (1995). Representation of spatial orientation by the intrinsic dynamics of the head-direction cell ensemble: A theory. *Submitted.*
